# Learning Plaintext-Ciphertext Cryptographic Problems via ANF-based SAT Instance Representation

**Xinhao Zheng, Yang Li, Cunxin Fan, Huaijin Wu, Xinhao Song, Junchi Yan**[‡]
Dept. of CSE & School of AI & Moe Key Lab of AI, Shanghai Jiao Tong University
{void_zxh, yanglily, sjtu18017295729,
whj1201, sxh001, yanjunchi}@sjtu.edu.cn

## Abstract

Cryptographic problems, operating within binary variable spaces, can be routinely transformed into Boolean Satisfiability (SAT) problems regarding specific cryptographic conditions like plaintext-ciphertext matching. With the fast development of learning for discrete data, this SAT representation also facilitates the utilization of machine-learning approaches with the hope of automatically capturing patterns and strategies inherent in cryptographic structures in a data-driven manner. Existing neural SAT solvers consistently adopt conjunctive normal form (CNF) for instance representation, which in the cryptographic context can lead to scale explosion and a loss of high-level semantics. In particular, extensively used XOR operations in cryptographic problems can incur an exponential number of clauses. In this paper, we propose a graph structure based on Arithmetic Normal Form (ANF) to efficiently handle the XOR operation bottleneck. Additionally, we design an encoding method for AND operations in these ANF-based graphs, demonstrating improved efficiency over alternative general graph forms for SAT. We then propose CryptoANFNet, a graph learning approach that trains a classifier based on a message-passing scheme to predict plaintext-ciphertext satisfiability. Using ANF-based SAT instances, CryptoANFNet demonstrates superior scalability and can naturally capture higher-order operational information. Empirically, CryptoANFNet achieves a 50x speedup over heuristic solvers and outperforms SOTA learning-based SAT solver NeuroSAT, with 96% vs. 91% accuracy on small-scale and 72% vs. 55% on large-scale datasets from real encryption algorithms. We also introduce a key-solving algorithm that simplifies ANF-based SAT instances from plaintext and ciphertext, enhancing key decryption accuracy from 76.5% to 82% and from 72% to 75% for datasets generated from two real encryption algorithms.

## 1 Introduction

Machine Learning (ML) has shown promise in solving discrete problems such as Mixed Integer Programming (MIP) [1] and Boolean Satisfiability (SAT) [2]. Meanwhile, ML has also made significant strides in the field of SAT-based cryptanalysis [3, 4]. Neural solvers for SAT [5, 6, 7] enjoy flexibility and automation in handling large datasets through data-driven learning, demonstrating advantages in prediction speed and adaptive learning. Plaintext-ciphertext cryptographic problems running in binary variable spaces can seamlessly be transformed into Boolean Satisfiability (SAT) problems, involving specific cryptographic conditions like plaintext-ciphertext matching. This SAT representation facilitates the utilization of machine learning methods to automatically learn inherent patterns and strategies in cryptographic structures in a data-driven manner.

---

[‡]Corresponding author. This work was supported by NSFC (92370201, 62222607) and Shanghai Municipal Science and Technology Major Project under Grant 2021SHZDZX0102.

However, the large scale and complex structure of cryptographic instances pose significant challenges for previous ML algorithms, which typically rely on conjunctive normal form (CNF) to encode cryptographic problems and are primarily effective with instances containing relatively small numbers of variables, usually in the range of tens. Cryptographic algorithms frequently entail constraints like XOR, modular addition, AND, and OR operations, thus making the logical XOR operator prevalent in SAT-based cryptanalysis and resulting in the proliferation of XOR clauses. However, XOR operations are generally challenging to represent in CNF. Converting a clause connected by XOR with $k$ literals into CNF is a well-known process that results in nearly $2^{k-1}$ OR clauses, each with $k$ literals. This could lead to the problem size ballooning considerably, thereby increasing computational complexity, and such encoding may compromise high-order operational information.

CNF is a conjunction (and-ing) of clauses, with each clause consisting of a disjunction (or-ing) of positive and negative variables called literals. The innate versatility of "and" and "or" constraint set of CNF allows for efficient application in SAT solving, yet it often requires introducing additional variables to represent more complex logical operations like XOR. Therefore, we propose to adopt an alternative constraint set, Algebraic Normal Form (ANF), to directly represent native XOR operations in the formulas. In ANF, formulas are expressed as collections of polynomials using addition (XOR) and multiplication (AND) operations on $\mathbb{GF}(2)$. Compared to CNFs, ANFs are more adept at capturing information about higher-order operations and are more concise in handling complex operations. This renders ANF a fitting choice for representing the structure and logic of cryptographic algorithms. However, representing ANF as graphs for efficient learning is challenging, and its application to existing SAT solvers is not as straightforward as CNF, resulting in obstacles to applying ANF in subsequent applications in complex scenarios like cryptographic problems.

To organize effective learning-based SAT practices in ANF, we propose an ANF-based graph structure to represent SAT instances for cryptographic applications. Building upon the graph representations, we propose CryptoANFNet, which employs a message-passing scheme and is trained as a classifier to predict satisfiability through dedicatedly constructed data encoded from cryptographic problems. The introduction of higher-order operations via ANF simplifies the resulting graph problems that would otherwise require hundreds or thousands of nodes using CNF to merely dozens of nodes. This simplification not only enhances the effectiveness of learning-based methods but also enables models to naturally capture higher-order operational information. Compared to learning-based SAT solvers, CryptoANFNet can handle larger instances with greater accuracy and achieves a 50x speedup over traditional heuristic solvers. Moreover, for the key-solving problem, we propose a key decryption algorithm that creates and simplifies ANF-based SAT instances from the plaintext and ciphertext, and uses the outputs from two derived SAT instances to deduce the correct key bit values. Given an input plaintext-ciphertext pair and an encryption algorithm, we proceed in three steps: 1) Use the encryption algorithm and the given plaintext-ciphertext to obtain the original ANF-based SAT instance; 2) For a specific bit in the key, create two extended SAT instances by assigning 0 or 1 to the chosen bit; 3) Pass these instances to CryptoANFNet in corresponding pairs, output paired scores, and determine the bit value based on which instance has a higher score. In this way, we can further improve the accuracy on datasets generated from two real encryption algorithms. **The highlights of this paper include:**

- Based on the Arithmetic Normal Form (ANF), we propose a graph structure to succinctly represent the excessive XOR operations in cryptographic problems. We then design two ways to encode the AND operations in the ANF-based graph to represent SAT instances derived from cryptographic problems. In Tab. 1, our ANF-graph is more efficient than the general graph form for SAT in [5].
- We propose (supervised) learning to solve (for the first time to our knowledge) the challenging cryptographic problem: plaintext-ciphertext satisfiability prediction, which could otherwise be intractable by traditional SAT methods. Our proposed GNN-based classifier CryptoANFNet with ANF-based SAT instances as input, achieves a 50x speedup over heuristic solvers, and outperforms the SOTA learning-based SAT solver NeuroSAT [5] by 96% vs. 91% and 72% vs. 55% accuracy on small and large scale datasets, generated from real encryption algorithms, respectively.
- We extend to the key decryption problem. We propose a key-solving algorithm that derives ANF-based SAT instances as further simplified by our devised techniques, from the plaintext and ciphertext, and use the output of two derived SAT instances to infer the key values. It boosts accuracy by 76.5%->82% and 72%->75% on datasets generated from two real encryption algorithms.

## 2 Related Works

**Learning for SAT.** Learning has shown promise in automatically uncovering heuristics for solving combinatorial problems in a data-driven manner [8, 9, 10, 11, 12, 13]. There are primarily two routes dominating these efforts in SAT solving. Learning-aided solvers replace certain components in traditional solvers with learning-aided counterparts to enhance performance. The learning utilization encompasses branching heuristics [14, 15], variable initialization [16, 17, 18], glue clause prediction [19], etc. End-to-end neural solvers conceive SAT solving as a graph-based prediction task based on its CNF representation. They utilize neural networks to directly predict satisfiability or minimal unsatisfied cores via graph massage-passing networks [5, 6] or transformers [20]. These methods do not rely on handcrafted strategies and demonstrate the potential for magnitude speedup by circumventing exhaustive search procedures. However, neural networks may struggle with large-scale problems, posing a significant challenge, particularly in cryptographic applications where the XOR operation can lead to scale explosion in CNF-based literal-clause graph representations. Besides solving, there are also learning-based efforts that provide support for problem solving, like generating pseudo-industrial instances to resolve data bottlenecks [21, 22, 23, 11].

**Learning for cryptanalysis.** The rise of deep learning has dramatically transformed the ability to analyze encrypted data in its raw state [24, 25], particularly for lightweight block ciphers, as noted in recent studies [26, 27, 28, 29]. Gohr [30] pioneered the application of learning-based cryptanalysis on round-reduced SPECK, demonstrating that deep differential-neural distinguishers could identify features that elude traditional strong distinguishers. This breakthrough highlighted the potential of neural networks in cryptanalysis. Subsequently, further insights into differential-neural distinguishers were provided by [31], showcasing that these networks could learn not only the differential distribution on the output pairs but also those in the penultimate and ante-penultimate rounds. Moreover, [32] developed a machine learning-based key-recovery attack that successfully carried out the first practical 13-round attacks on SPECK, with extensions to 14-rounds subsequently reported in [33]. However, despite the introduction of neural networks, these methods still fall within the framework of differential analysis and do not directly utilize machine learning to handle plaintext-ciphertext satisfiability prediction or key-solving problems.

**SAT for cryptanalysis** The cryptography community has significantly increased its use of automated tools for cryptanalysis, specifically in searching for linear and differential trails, leveraging methods such as the Boolean Satisfiability Problem (SAT)[34, 35, 4, 36] and Mixed Integer Linear Programming (MILP)[37]. In SAT-based cryptanalysis, significant strides have been made: [38] explored differential trails in ARX ciphers using SAT methods, while [39] developed an automated search technique for ciphers with Sboxes to achieve more accurate and probable differential trails. Additionally, [40] created an SAT-based automated search toolkit applied to SHA-3. These efforts essentially involve converting cryptographic algorithms into SAT instances and then conducting cryptanalysis by searching for differential paths, but they all need handcrafted strategies.

To date, no methods have focused on establishing effective graph representations and developing SAT solvers based on the XOR-friendly ANF representation. This paper pioneers the utilization of graph forms to represent the excessive XOR operations in cryptographic problems succinctly. As far as we know, we are the first to integrate graph neural networks and SAT for encryption algorithms.

## 3 Preliminaries

**ANF formula.** Algebraic Normal Form (ANF) is a polynomial representation of Boolean functions, instrumental in cryptographic applications due to its efficient handling of XOR and AND operations. Formally, a logical formula can be rewritten as a boolean function: $\mathbb{GF}_2$: multiplication in $\mathbb{GF}_2(\cdot)$ becomes the AND operation and addition in $\mathbb{GF}_2(+)$ becomes the XOR operation. Then, the SAT instance constituted by the conjunction of logical formulas can be rewritten as several Boolean functions equal to 0, *i.e.*, several Boolean equations. Therefore, a standard SAT instance in ANF form can be described by multiple Boolean equations, as illustrated by the example of an ANF-form SAT instance shown at the top of Fig. 1 (a).

Here, each variable is referred to as a vanilla literal. In a Boolean equation, the right side is always 0 and the Boolean function on the left side, formed by the XOR connection of monomials, is called a clause. Each monomial is either a constant term 1, a vanilla literal, or a product of variables. The

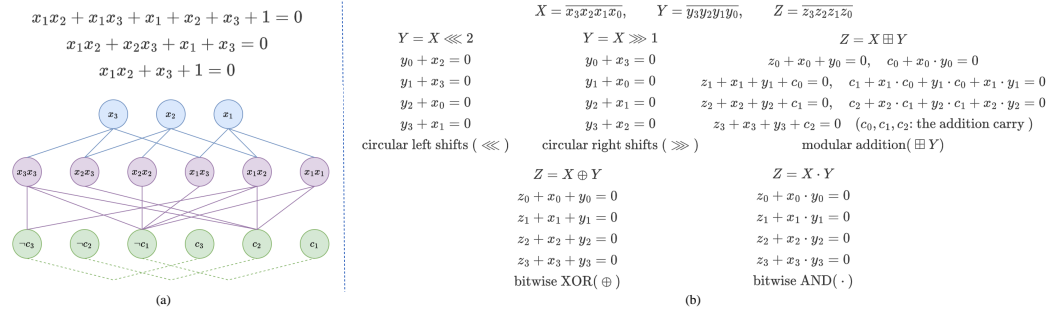

Figure 1: (a) Example ANF formula graph from MQ problem; (b) The transformations to express the circular left shifts ($\lll$), circular right shifts ($\ggg$), modular addition ($\boxplus$), bitwise XOR ($\oplus$), and bitwise AND ($\cdot$) operations in ANF.

product of variables is called a literal of different orders based on the number of variables multiplied; for example, $x_1 x_2$ is called a second-order literal. A literal appears in a clause if and only if that literal exists in the Boolean function represented by the clause. For simplicity, we will omit the multiplication operator ($\cdot$) for the rest of the paper - as long as its use in monomials is implicit.

**MQ problem.** Encryption algorithms, like SPECK, typically use a set of simple and efficient operations, including circular left shifts ($\lll$), circular right shifts ($\ggg$), modular addition ($\boxplus$), bitwise XOR ($\oplus$), and bitwise AND ($\cdot$) operations. The algebraic properties of these operations allow them to be converted into systems of boolean polynomial equations. In the examples shown in Fig. 1(b), circular shifts, XOR, and bitwise AND operations can be directly expressed in polynomial form, while modular addition can be represented through quadratic equations. These equations consist solely of variables, second-order products of variables, and the Boolean constant term, which means the encryption process can be transformed into a Multivariate Quadratic (MQ) problem.

In practice, for example, consider the toy formula for the 8-bit plaintext in one round shown below:

$$X = \overline{x_3 x_2 x_1 x_0}, \qquad Y = \overline{y_3 y_2 y_1 y_0}, \qquad K = \overline{k_3 k_2 k_1 k_0}$$
$$Z = \overline{z_3 z_2 z_1 z_0}, \qquad W = \overline{w_3 w_2 w_1 w_0}$$
$$Z = ((X \ggg 2) \boxplus Y) \oplus K, \qquad W = (Y \lll 1) \cdot X$$

where $K$ represents the key for the encryption algorithm, $X$ and $Y$ represent the high and low bits of the round function's input, respectively, and $Z$ and $W$ represent the high and low bits of the round function's output. It can be transformed into several standard multivariate quadratic equations:

$$x_2 + y_0 + s_0 = 0, \quad x_2 y_0 + c_0 = 0, \quad x_3 + y_1 + c_0 + s_1 = 0, \quad x_3 c_0 + y_1 c_0 + x_3 y_1 + c_1 = 0$$
$$x_0 + y_2 + c_1 + s_2 = 0, \quad x_0 c_1 + y_2 c_1 + x_0 y_2 + c_2 = 0, \quad x_1 + y_3 + c_2 + s_3 = 0$$
$$z_3 + s_3 + k_3 = 0, \quad z_2 + s_2 + k_2 = 0, \quad z_1 + s_1 + k_1 = 0, \quad z_0 + s_0 + k_0 = 0$$
$$w_3 + x_3 y_2 = 0, \quad w_2 + x_2 y_1 = 0, \quad w_1 + x_1 y_0 = 0, \quad w_0 + x_0 y_3 = 0$$

where $x_i$, $y_i$, $z_i$, $w_i$, $s_i$, and $c_i$ are all Boolean variables and each equation has its right side equal to 0. Thus, by using these transformations, we can convert the encryption process into an MQ problem in ANF, and in the following section, we will only address solving the MQ problem.

# 4 Model

## 4.1 Approach Overview

**ANF-based Graph Structures** Given the ANF formula of an MQ problem, we propose a graph-based representation for ANF. First, we categorize clause nodes into positive and negative based on whether they contain a constant term. We then directly encode ANF formula into an undirected graph for representing variables and second-order literals: We encode the ANF as an undirected graph with one node for every independent literal and second-order literal, two complementary nodes for every clause with the same set of literals, an edge between every literal and every second-order literal

it corresponds to, an edge between every second-order literal and every clause it appears in, and a different type of edge between each pair of complementary clauses.

**General Architecture** Similar to NeuroSAT [5], we employ a message-passing neural network to derive clause and literal embeddings by executing a specified number of iterations. Then, we use these embeddings to predict a satisfiability vote through classification multilayer perceptrons(MLPs). Note that we have two key differences: 1) We only retain embeddings for vanilla literals and clauses. The second-order literals' embeddings are derived by concatenating first-order literals's embeddings. 2) We only use clause embeddings and apply two separate classification MLPs for positive and negative clause embeddings to predict the satisfiability vote, respectively.

**Key-solving Algorithm** With the model CryptoANFNet, we can evaluate the satisfiability of an individual ANF instance individual. Due to the uniqueness of solutions in key-solving problems, we propose an algorithm to determine the value of a specific bit in a key. Given a plaintext-ciphertext pair and an encryption algorithm, our approach derives an ANF-based SAT instance from the input data, generates two derived SAT instances for a specific bit by assigning 0 or 1, and then uses a scoring model to evaluate and determine the most likely value for the bit.

### 4.2 ANF formula Graph

Similar to CNF, the literal-clause structure in an ANF formula has inherent permutation invariance, making it compatible with graph-based representations. However, unlike in CNF, there are no negated literals in ANF. Instead, due to the constant term, there are complementary relationships between clauses that contain the same literals but have different constant terms. Additionally, compared to CNF, the clauses in ANF formulas exhibit internal asymmetry, where high-order literals, created by combining vanilla literals with AND operations, form a special structure within the clause and require further decomposition. Note that in the MQ problem, the ANF formula has at most second-order literals. To represent second-order literals, we directly encode an ANF into an undirected graph $G$, which can directly be derived from an ANF formula as follows:

- Each vanilla literal $x_i$ and second-order literal $x_i x_j$ (including $x_i x_i$) in the ANF becomes a vertex in graph $G$.
- Each clause becomes two vertices ($\bar{c}_i$ and $c_i$) in graph $G$ according to the set of literals it contains, where $c_i$ denotes clauses with constant term 0, called the positive clause, while $\bar{c}_i$ denotes clauses with constant term 1, called the negative clause.
- An edge between second-order literal $x_i x_j$ and clause $c_k$ is in $g$ if and only if $x_i x_j$ appears in clause $c_k$. Especially, an edge between second-order literal $x_i x_i$ and clause $c_k$ is in $G$ if and only if $x_i$ appears in clause $c_k$.
- A different edge between each clause vertex $c_i$ and its complementary clause vertex $\bar{c}_i$.
- A special edge between vanilla literal $x_i$ (and $x_j$) and second-order literal $x_i x_j$ is in $G$.

Fig. 1 (a) shows an example graph derived from an ANF formula. The graph is a tripartite graph with three columns of nodes from top to bottom: vanilla literals, second-order literals, and clauses. Nodes at each level are connected to related nodes at the next level, and clauses are additionally connected to their complementary clauses. Besides, we introduce another graph structure that transforms ANF into a bipartite graph by replacing second-order literals with independent variables. This facilitates the use of networks designed for the CNF formula but introduces more nodes. Please refer to Appendix B for more details.

### 4.3 Model Architecture

Given the ANF formula graph, we propose the neural network model, CryptoANFNet, which first extracts the embedding of vanilla literals and clauses from the ANF graph and then predicts the satisfiability vote based on the embedding of clauses. CryptoANFNet uses a message-passing neural network to iteratively refine a vector space embedding for nodes in the graph. Here, different from NeuroSAT, at each time step $t$, we only save the embedding of vanilla literals and pairs of complementary clauses.

In each iteration, the model aggregates information from neighboring nodes to update the embeddings. First, each clause receives information from associated literals and its complementary clause and updates its embedding accordingly. We obtain the second-order literal embedding by concatenating

the vanilla literals. Each clause then receives messages from neighboring second-order literal nodes and its complementary clause based on the graph's connectivity. Next, each literal updates its embedding by receiving messages from associated clauses. Specifically, this information transfer occurs in two rounds: first from the clause to the intermediate second-order literal and then from the second-order literal to the vanilla literal.

Formally, at every time step $t$, we have a matrix $L^{(t)} \in \mathbb{R}^{n \times d}$ whose $i$-th row contains the embedding for the vanilla literal $l_i$ and two matrices $C_{\text{pos}}^{(t)}, C_{\text{neg}}^{(t)} \in \mathbb{R}^{m \times d}$ whose $j$-th row contains the embedding for the positive clause $c_j$ and the relevant negative clause $\bar{c}_j$. We initialize these matrices by tiling $L^{(0)}, C_{\text{pos}}^{(0)}, C_{\text{neg}}^{(0)} \in \mathbb{R}^d$, respectively. Then, a single iteration consists of the following two updates:

First, each clause receives messages from associated literals and its complementary clause and updates its embedding accordingly.

$$
\begin{aligned}
L_{l2c}^{(t)} =& L_{l2l}([L^{(t)}[I], L^{(t)}[J]]) \\
[C_{m,\text{pos}}^{(t)}, C_{m,\text{neg}}^{(t)}] =& M_{l2c}^T L_{\text{msg}}(L_{l2c}^{(t)}) \\
(C_{\text{pos}}^{(t+1)}, C_{h,\text{pos}}^{(t+1)}) \leftarrow& C_{u,\text{pos}}([C_{h,\text{pos}}^{(t)}, C_{\text{neg}}^{(t)}, C_{m,\text{pos}}^{(t)}]) \\
(C_{\text{pos}}^{(t+1)}, C_{h,\text{neg}}^{(t+1)}) \leftarrow& C_{u,\text{neg}}([C_{h,\text{neg}}^{(t)}, C_{\text{pos}}^{(t)}, C_{m,\text{neg}}^{(t)}])
\end{aligned}
\tag{1}
$$

where $I = [0, 1, \cdots, n-1, 0, 1, \cdots, n-2, n-1] \in \mathbb{R}^{n^2}$ and $J = [0, 0, \cdots, 0, 1, 1, \cdots, n-1, n-1] \in \mathbb{R}^{n^2}$ represents the row and column index respectively. $M_{l2c}$ is the adjacency matrix defined by $M_{l2c}(i,j) = \mathbf{1}$(the $i$-th second-order literal appears in the $j$-th clause). $L_{l2l}, L_{\text{msg}}$ are two MLPs and $C_{u,\text{pos}}, C_{u,\text{neg}}$ are two layer-norm LSTMs [41] with hidden states $C_{h,\text{pos}}^{(t)}, C_{h,\text{neg}}^{(t)} \in \mathbb{R}^{m \times d}$ respectively.

Next, each literal updates its embedding by receiving messages from associated clauses.

$$
L_{c2l}^{(t)} = M_{l2c} C_{\text{msg}}([C_{\text{pos}}^{(t)}, C_{\text{neg}}^{(t)}])
\tag{2}
$$

$$
L_m^{(t)} = M_{l2l} L_{l2m}(L_{c2l}^{(t)})
\tag{3}
$$

$$
(L^{(t+1)}, L_h^{(t+1)}) \leftarrow L_u([L_h^{(t)}, L_m^{(t)}])
\tag{4}
$$

where $M_{l2l}$ is the the adjacency matrix defined by $M_{l2l}(i,j) = \mathbf{1}$ (the $i$-th vanilla literal appears in the $j$-th second-order literal). $L_{l2m}, C_{\text{msg}}$ are two MLPs and $L_u$ is a layer-norm LSTMs with hidden states $L_h^{(t)} \in \mathbb{R}^{n \times d}$.

After $T$ iterations for updating clause and literal embeddings, we use these embeddings to predict the satisfiability of the ANF formula. This prediction process involves a feedforward neural network that combines the embeddings of clauses to output a satisfiability vote representing whether the formula is satisfiable or not. The feedforward neural network consists of two MLPs, which are designed for the positive and negative clauses respectively, and they compute each clause's satisfiability score for the SAT instance based on its embedding. Then we sum up the votes of all the clauses and output the final satisfiability vote. This architecture is designed to be flexible and scalable, accommodating various ANF graph sizes and complexities.

Formally, the feedforward neural network outputs the satisfiability vote based on the clauses's embedding ($C_{\text{pos}}, C_{\text{neg}} \in \mathbb{R}^{m \times d}$) as follows:

$$
\begin{aligned}
V_* \leftarrow& C_{\text{vote}}^1(C_{\text{pos}}), \ \ \overline{V}_* \leftarrow C_{\text{vote}}^2(C_{\text{neg}}) \\
& s \leftarrow \sigma(\text{sum}(V_* + \overline{V}_*))
\end{aligned}
\tag{5}
$$

where $V_*, \overline{V}_* \in \mathbb{R}^{m \times 1}$ represent the satisfiability score of positive and negative clauses, respectively. $C_{\text{vote}}^1, C_{\text{vote}}^2$ denote two MLPs. $\text{sum}(V_* + \overline{V}_*)) \in \mathbb{R}$ represent the sum of the clause votes, and $\sigma(\cdot)$ represent the sigmoid funtion. $s \in [0, 1]$ represents the output prediction. We train the network to minimize the binary cross-entropy loss between the output prediction $s$ and the true label $y$.

### 4.4 Key-solving Algorithm

While we train CryptoANFNet as a classifier to predict satisfiability, we can also use CryptoANFNet for key solving. Given a plaintext-ciphertext pair and an encryption algorithm, we can transform

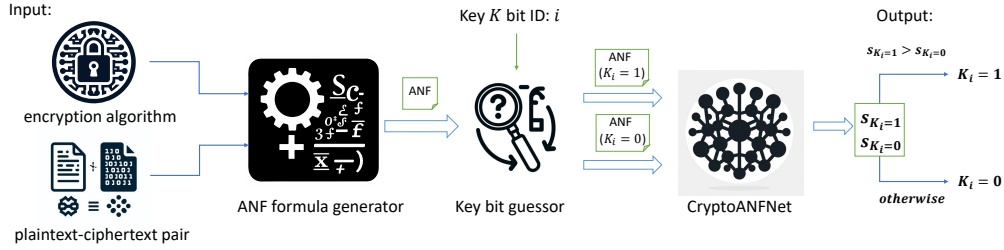

Figure 2: **The pipeline of the key-solving algorithm.** Given a plaintext-ciphertext pair and an encryption algorithm, we first transform them into an ANF-based instance of an MQ problem. Then, for a specific key bit $K_i$ (the $i$-th bit of key), we guess its value as either $0$ or $1$ and generate two derived SAT instances. We then employ CryptoANFNet to predict the satisfiability of each instance. The final determination of $K_i$ is based on which instance receives a higher satisfiability score.

them into an ANF-based instance of an MQ problem. Since key solutions are typically unique, a natural idea is to assign values to each bit of the key one by one, derive extended instances from the original ANF instance, and then use CryptoANFNet's prediction results to determine each bit's value. For example, consider a 4-bit key $K = \overline{k_3 k_2 k_1 k_0}$. We could guess $K = \overline{1 k_2 k_1 k_0}$ to get a derived ANF instance and then let CryptoANFNet predict whether the derived ANF instance is satisfiable. If it is, we conclude $k_3 = 1$ otherwise, $k_3 = 0$.

In essence, it is not necessary to obtain exact classification probabilities for individual samples. Instead, we only need to compare the satisfiability votes for samples generated by making two different guesses for the same bit. Based on this idea, we propose the key-solving algorithm to determine the value of a specific bit in the key. The algorithm consists of three steps. First, we obtain the original ANF-based SAT instance from the encryption algorithm and the plaintext-ciphertext pair. Second, for a specific key bit, we create two derived SAT instances by assigning either 0 or 1 to the selected bit. Third, we use CryptoANFNet to predict the satisfiability votes of these two instances and determine the bit's value based on which instance scores higher. Fig. 2 provides a high-level illustration of the algorithm.

Besides, in this algorithm, we use the derived SAT instances in pairs to train the network, CryptoANFNet, by minimizing the loss function, as follows.

$$L_{\text{cls}} = \sum_i^{2N} \text{BCE}(\sigma(s_i), y_i), \quad L_{\text{comp}} = \sum_i^{N} \text{CE}(\text{softmax}([s_{2i}, s_{2i+1}]), [y_{2i}, y_{2i+1}]) \quad (6)$$

$$L_{\text{final}} = L_{\text{cls}} + \lambda \times L_{\text{comp}} \quad (7)$$

where $s_i \in \mathbb{R}$ denotes the satisfiability vote of the $i$-th instance, $y_i \in \{0, 1\}$ denotes the corresponding label. $BCE(\cdot)$, $CE(\cdot)$, $\sigma(\cdot)$ and softmax$(\cdot)$ represent the binary cross-entropy loss, the cross-entropy loss, the sigmoid function, and the softmax function. Especially, $s_{2i}$ and $s_{2i+1}$ are two corresponding votes for derived SAT instances, generated by assigning either 0 or 1 to the selected bit of the key in the same original ANF instance.

## 5 Experiments

### 5.1 Datasets

For both training and test datasets, we evaluate our approach on two types of synthetic datasets.

For the first synthetic dataset, we use a similar approach to NeuroSAT to generate instances of the MQ problem. These instances consist of the dataset called SR(n), where $n$ is the number of variables in the instance. The distribution SR(n) contains pairs of $n$ variables of stochastic SAT instances in the ANF formula. The pair of instances have the following properties: one of them is satisfiable and the other is unsatisfiable, and the difference between the two instances is only in the constant term in one of the clauses. To generate a clause $c_i$ of an SAT instance in ANF with $n$ variables, we randomly sample a number $k$ ($3 \leq k \leq 2n$) as the clause's length. Then, from all $n^2$ second-order polynomials,

Table 1: Parameters of SAT problems in CNF and ANF

|  | Datasets | SR(5) | SR(25) | Scipher 3-8-16 | Scipher 3-16-32 | Scipher 6-8-16 | Scipher 6-16-32 | Speck 3-8-16 | Speck 6-8-16 |
|---|---|---|---|---|---|---|---|---|---|
| CNF | #Literals | 6 | 424 | 25 | 49 | 49 | 97 | 57 | 129 |
| | #Clauses | 75 | 5492 | 195 | 225 | 735 | 1519 | 336 | 921 |
| | #Nodes | 87 | 6340 | 245 | 323 | 833 | 1713 | 450 | 1179 |
| ANF | #Literals | 5 | 25 | 24 | 48 | 48 | 96 | 56 | 128 |
| | #Clauses | 11 | 26 | 24 | 48 | 48 | 96 | 64 | 136 |
| | #Nodes | 27 | 77 | 72 | 144 | 144 | 288 | 184 | 400 |

Table 2: Performance of different learning-based solvers on synthetic datasets

| Datasets | SR(5) | SR(25) | Scipher 3-8-16 | Scipher 3-16-32 | Scipher 6-8-16 | Scipher 6-16-32 | Speck 3-8-16 | Speck 6-8-16 |
|---|---|---|---|---|---|---|---|---|
| NeuroSAT | 91.0% | 57.0% | 74.0% | 72.7% | 53.0% | 51.0% | 55.0% | 52.5% |
| CryptoANFNet | **96.0%** | **72.0%** | **76.5%** | **75.6%** | **69.0%** | **66.5%** | **72.0%** | **68.5%** |

we sample $k$ to include as literals in the clause. Polynomials of the form $x_i^2$ are regarded as producing vanilla literal $x_i$, while those of the form $x_i x_j (i \neq j)$ are considered as producing second-order literal $x_i x_j$. Additionally, there's a 50% probability of adding a constant term of 1 to the clause. Based on this, in the approach, we continue generating clauses adding them to the SAT instance, and then using a traditional ANF-accessible solver, WDSat [42], to check the satisfiability until adding clause $c_m$ finally made the instance unsatisfiable.

For the second synthetic dataset, we utilize instances generated from the real encryption algorithms to construct the dataset. Specifically, we use a lightweight and popular block cipher (for certain reasons, we do not disclose the specific name of the encryption algorithm, referred to as Scipher; for its detailed encryption process, please refer to Appendix C) and the SPECK algorithm. Given the length $k$ of the seed key, the length $n$ of the plaintext, and the number of encryption rounds $r$, we randomly generate the seed key and plaintext to generate an SAT instance in ANF corresponding to the encryption algorithm. Then, in each round, we use the seed key to encrypt the plaintext and obtain the final ciphertext. Based on the generated plaintext-ciphertext pairs and the encryption process, we obtain the original ANF instance. Then, we select a random bit of the seed key and modify it with 0 and 1 to obtain the corresponding satisfiable and unsatisfiable instances, thus forming the final dataset. Depending on the differences in the encryption algorithms used, the datasets generated by this process are called Scipher-r-k-n and Speck-r-k-n.

## 5.2 Satisfiability Prediction

**Complexity of SAT instances.** To compare the solving efficiency of CNF solvers and ANF solvers, we first conduct complexity comparison experiments on SAT instances. By representing the same data set using CNF and ANF, we compare the number of variables, the number of clauses, and the number of parameterized nodes in the constructed graphs in the learning-based solvers. Table 1 lists the parameters of SAT instances for each dataset under CNF and ANF. Our ANF graph is more efficient than the CNF graph, which is a general graph previously used to represent SAT problems. In datasets containing XOR operations, CNF typically requires transforming logical expressions into conjunctive normal form, which may increase the number of literals and clauses; whereas ANF directly represents logical functions using polynomials, thus potentially having fewer literals.

**Evaluation on different datasets.** To evaluate the performance of our proposed CryptoANFNet, we conduct a series of experiments comparing it against the state-of-the-art model, NeuroSAT [5]. The primary focus of these experiments is solving SAT instances derived from the real encryption algorithms. For each dataset, we use Bosphorus [43] to convert (without simplification) the ANF-based SAT instances in our dataset into CNF-based SAT instances, resulting in a CNF form of the same dataset. We then train CryptoANFNet on the CNF-form dataset and compared its results with those of NeuroSAT, which was trained on the ANF-form dataset. Table 2 lists the results of

Table 3: Performance for key-solving algorithm in solving MQ problems on synthetic datasets.

| Datasets | Scipher 3-8-16 | Scipher 3-16-32 | Scipher 6-8-16 | Scipher 6-16-32 | Speck 3-8-16 | Speck 6-8-16 |
|---|---|---|---|---|---|---|
| NeuroSAT | 74.0% | 72.7% | 53.0% | 51.0% | 55.0% | 52.5% |
| CryptoANFNet | 76.5% | 75.6% | 69.0% | 66.5% | 72.0% | 68.5% |
| CryptoANFNet+ key-solving | **82.0%** | **78.4%** | **70.0%** | **69.0%** | **75.0%** | **71.0%** |

Table 4: Comparing the efficiency of different solvers for solving the MQ problem on synthetic datasets. (Average runtime: (SAT, UNSAT) ms/instance)

| Datasets | SR(5) | SR(25) | Scipher 3-8-16 | Scipher 3-16-32 | Scipher 6-8-16 | Scipher 6-16-32 | Speck 3-8-16 | Speck 6-8-16 |
|---|---|---|---|---|---|---|---|---|
| NeuroSAT [5] | (3,3) | (20,20) | (7,7) | (10,10) | (7,7) | (14,14) | (13,13) | (18,18) |
| CryptoANFNet | (2,2) | (5,5) | (8,8) | (9,9) | (10,10) | (8,8) | (11,11) | (14,14) |
| WDSat [42] | (36,34) | (2470,5662) | (38,38) | (39,39) | (40,37) | (86,150) | (44,47) | (46,46) |
| CryptoMiniSat [44] | (4,4) | (13491,35912) | (4,4) | (7,9) | (8,9) | (410,1354) | (5,5) | (6,8) |
| Kissat [45] | (2,2) | (4922,14856) | (2,2) | (2,2) | (5,8) | (219,464) | (3,3) | (4,5) |

NeuroSAT and CryptoANFNet on synthetic datasets of different scales. On small-scale datasets, both models achieve high accuracy, but CryptoANFNet still outperforms NeuroSAT, such as 96% vs. 91% accuracy on the SR(5) dataset. On large-scale datasets, like SR(25), NeuroSAT can hardly predict the satisfiability of instances, with its prediction accuracy hovering around 50% and CryptoANFNet significantly outperforms NeuroSAT, such as 72% vs. 57% accuracy on the SR(25) dataset and 72% vs. 55% accuracy on the Speck-3-8-16 dataset.

## 5.3 Key Bit Prediction

In addition to evaluating the satisfiability of individual ANF instances with the model CryptoANFNet, we conduct a comparative experiment to validate the key-solving algorithm. We assess the effectiveness of the key-solving algorithm in determining specific bit values in cryptographic keys by testing on datasets generated from real encryption algorithms, such as Scipher-r-k-n and Speck-r-k-n. For each dataset, we use the associated SAT-UNSAT instance pairs generated by the key-solving algorithm to train CryptoANFNet. In this experiment, we set $\lambda = 0.1$. Table 3 lists the comparison of results before and after using the key-solving algorithm. As shown, the key-solving algorithm somewhat improves performance on the tested datasets. Notably, it boosts the accuracy from 76.5% to 82% and from 72% to 75% on the Scipher-3-8-16 and Speck-3-8-16 datasets, respectively.

## 5.4 Comparsion with Heuristic SAT solvers

In this section, we evaluate the efficiency of our proposed model, CryptoANFNet, in comparison with traditional heuristic SAT solvers. These experiments focused on assessing the efficiency of solving SAT instances derived from real encryption algorithms. We compare CryptoANFNet to the following solvers: the best currently available implementation of CryptoMiniSat [44] (a CNF-based solver specifically designed for handling complex problems in cryptanalysis), Kissat [45] (a highly efficient CNF-based solver that has demonstrated its ability to solve challenging SAT instances in the SAT competition), and WDSat [42] (an ANF-based solver for instances in algebraic normal form).

To fairly compare the efficiency of solvers, we test the unit solving time (ms/instance) for UNSAT and SAT instances for each dataset. For the ANF-based solver, we directly test the instances in ANF form. For CNF-based solvers, we use Bosphorus [43] to convert the ANF instances (without simplification) to CNF form for testing. We test all solvers on an AMD Ryzen Threadripper 3970X 32-Core Processor and an NVIDIA GeForce RTX 3090 GPU. Table 4 lists the results of different solvers and more results are shown in Appendix A. We find that the learning-based model CryptoANFNet and NeuroSAT solve SAT instances derived from cryptographic problems much more quickly and CryptoANFNet achieves a 50x speedup on average over traditional heuristic solvers. These results make it possible to apply learning-based solvers in practice.

# 6 Conclusion and Outlook

In this paper, we propose a graph structure based on Arithmetic Normal Form (ANF) to efficiently handle XOR operations in cryptographic problems. We also design an encoding method for AND operations within these ANF-based graphs, demonstrating improved efficiency over traditional graph forms. Building on this, we introduce CryptoANFNet, a graph learning framework for SAT-based cryptanalysis utilizing ANF. CryptoANFNet addresses the challenging problem of plaintext-ciphertext satisfiability prediction and achieves a remarkable 50x speedup over heuristic solvers. It also outperforms the state-of-the-art learning-based SAT solver, NeuroSAT. Furthermore, we introduce a key-solving algorithm that simplifies ANF-based SAT instances derived from plaintext and ciphertext, enhancing key decryption accuracy. In this way, future research could explore further optimization of the ANF-based graph structures and the integration of more advanced neural network models to push the boundaries of cryptographic problem-solving.

# References

[1] J. Zhang, C. Liu, X. Li, H.-L. Zhen, M. Yuan, Y. Li, and J. Yan, "A survey for solving mixed integer programming via machine learning," *Neurocomputing*, vol. 519, pp. 205–217, 2023.

[2] W. Guo, H.-L. Zhen, X. Li, W. Luo, M. Yuan, Y. Jin, and J. Yan, "Machine learning methods in solving the boolean satisfiability problem," *Machine Intelligence Research*, 2023.

[3] L. Sun, D. Gerault, A. Benamira, and T. Peyrin, "Neurogift: Using a machine learning based sat solver for cryptanalysis," in *Cyber Security Cryptography and Machine Learning*, S. Dolev, V. Kolesnikov, S. Lodha, and G. Weiss, Eds. Cham: Springer International Publishing, 2020, pp. 62–84.

[4] S. Nejati and V. Ganesh, "Cdcl (crypto) sat solvers for cryptanalysis," *arXiv preprint arXiv:2005.13415*, 2020.

[5] D. Selsam, M. Lamm, B. Bünz, P. Liang, L. de Moura, and D. L. Dill, "Learning a sat solver from single-bit supervision," in *International Conference on Learning Representations*, 2019.

[6] Z. Li, J. Guo, and X. Si, "G4satbench: Benchmarking and advancing sat solving with graph neural networks," *arXiv preprint arXiv:2309.16941*, 2023.

[7] C. Cameron, R. Chen, J. Hartford, and K. Leyton-Brown, "Predicting propositional satisfiability via end-to-end learning," in *Proceedings of the AAAI Conference on Artificial Intelligence*, vol. 34, no. 04, 2020, pp. 3324–3331.

[8] Y. Bengio, A. Lodi, and A. Prouvost, "Machine learning for combinatorial optimization: a methodological tour d'horizon," *European Journal of Operational Research*, vol. 290, no. 2, pp. 405–421, 2021.

[9] Y. Li, J. Guo, R. Wang, and J. Yan, "T2t: From distribution learning in training to gradient search in testing for combinatorial optimization," in *Advances in Neural Information Processing Systems*, 2023.

[10] X. Li, F. Zhu, H.-L. Zhen, W. Luo, M. Lu, Y. Huang, Z. Fan, Z. Zhou, Y. Kuang, Z. Wang *et al.*, "Machine learning insides optverse ai solver: Design principles and applications," *arXiv preprint arXiv:2401.05960*, 2024.

[11] Z. Guo, Y. Li, C. Liu, W. Ouyang, and J. Yan, "Acm-milp: Adaptive constraint modification via grouping and selection for hardness-preserving milp instance generation," in *The Forty-first International Conference on Machine Learning*, 2024.

[12] H. Geng, H. Ruan, R. Wang, Y. Li, Y. Wang, L. Chen, and J. Yan, "There is no silver bullet: Benchmarking methods in predictive combinatorial optimization," *Advances in Neural Information Processing Systems*, 2024.

[13] Y. Li, J. Guo, R. Wang, H. Zha, and J. Yan, "Fast t2t: Optimization consistency speeds up diffusion-based training-to-testing solving for combinatorial optimization," in *Advances in Neural Information Processing Systems (NeurIPS)*, 2024.

[14] D. Selsam and N. Bjørner, "Guiding high-performance sat solvers with unsat-core predictions," in *International Conference on Theory and Applications of Satisfiability Testing*, 2019.

[15] V. Kurin, S. Godil, S. Whiteson, and B. Catanzaro, "Can q-learning with graph networks learn a generalizable branching heuristic for a sat solver?" *Advances in Neural Information Processing Systems*, vol. 33, pp. 9608–9621, 2020.

[16] H. Wu, "Improving sat-solving with machine learning," in *Proceedings of the 2017 ACM SIGCSE Technical Symposium on Computer Science Education*, 2017, pp. 787–788.

[17] H. Duan, S. Nejati, G. Trimponias, P. Poupart, and V. Ganesh, "Online bayesian moment matching based sat solver heuristics," in *International Conference on Machine Learning*. PMLR, 2020, pp. 2710–2719.

[18] Z. Li and X. Si, "Nsnet: A general neural probabilistic framework for satisfiability problems," *Advances in Neural Information Processing Systems*, vol. 35, pp. 25 573–25 585, 2022.

[19] J. M. Han, "Enhancing sat solvers with glue variable predictions," *arXiv preprint arXiv:2007.02559*, 2020.

[20] Z. Shi, M. Li, Y. Liu, S. Khan, J. Huang, H.-L. Zhen, M. Yuan, and Q. Xu, "Satformer: Transformer-based unsat core learning," in *2023 IEEE/ACM International Conference on Computer Aided Design (ICCAD)*. IEEE, 2023, pp. 1–4.

[21] J. You, H. Wu, C. Barrett, R. Ramanujan, and J. Leskovec, "G2sat: Learning to generate sat formulas," in *Advances in neural information processing systems*, vol. 32, 2019.

[22] Y. Li, X. Chen, W. Guo, X. Li, W. Luo, J. Huang, H.-L. Zhen, M. Yuan, and J. Yan, "Hardsatgen: Understanding the difficulty of hard sat formula generation and a strong structure-hardness-aware baseline," in *ACM SIGKDD International Conference on Knowledge Discovery and Data Mining (KDD)*, 2023.

[23] X. Chen, Y. Li, R. Wang, and J. Yan, "Mixsatgen: Learning graph mixing for sat instance generation," in *The Twelfth International Conference on Learning Representations*, 2024.

[24] E. Wenger, M. Chen, F. Charton, and K. E. Lauter, "Salsa: Attacking lattice cryptography with transformers," *Advances in Neural Information Processing Systems*, vol. 35, pp. 34 981–34 994, 2022.

[25] C. Li, E. Wenger, Z. Allen-Zhu, F. Charton, and K. E. Lauter, "Salsa verde: a machine learning attack on lwe with sparse small secrets," *Advances in Neural Information Processing Systems*, vol. 36, pp. 53 343–53 361, 2023.

[26] A. Singh, K. B. Sivangi, and A. N. Tentu, "Machine learning and cryptanalysis: An in-depth exploration of current practices and future potential," *Journal of Computing Theories and Applications*, vol. 2, no. 1, pp. 27–42, 2024.

[27] J. So, "Deep learning-based cryptanalysis of lightweight block ciphers," *Security and Communication Networks*, vol. 2020, pp. 1–11, 2020.

[28] A. Baksi and A. Baksi, "Machine learning-assisted differential distinguishers for lightweight ciphers," *Classical and Physical Security of Symmetric Key Cryptographic Algorithms*, pp. 141–162, 2022.

[29] Z. Hou, J. Ren, and S. Chen, "Cryptanalysis of round-reduced simon32 based on deep learning," *Cryptology ePrint Archive*, 2021.

[30] A. Gohr, "Improving attacks on round-reduced speck32/64 using deep learning," in *Advances in Cryptology–CRYPTO 2019: 39th Annual International Cryptology Conference, Santa Barbara, CA, USA, August 18–22, 2019, Proceedings, Part II 39*. Springer, 2019, pp. 150–179.

[31] A. Benamira, D. Gerault, T. Peyrin, and Q. Q. Tan, "A deeper look at machine learning-based cryptanalysis," in *Advances in Cryptology–EUROCRYPT 2021: 40th Annual International Conference on the Theory and Applications of Cryptographic Techniques, Zagreb, Croatia, October 17–21, 2021, Proceedings, Part I 40*. Springer, 2021, pp. 805–835.

[32] Z. Bao, J. Guo, M. Liu, L. Ma, and Y. Tu, "Enhancing differential-neural cryptanalysis," in *International Conference on the Theory and Application of Cryptology and Information Security*. Springer, 2022, pp. 318–347.

[33] Z. Bao, J. Lu, Y. Yao, and L. Zhang, "More insight on deep learning-aided cryptanalysis," in *International Conference on the Theory and Application of Cryptology and Information Security*. Springer, 2023, pp. 436–467.

[34] F. Lafitte, "Cryptosat: a tool for sat-based cryptanalysis," *IET Information Security*, vol. 12, no. 6, pp. 463–474, 2018.

[35] M. Soos, "The cryptominisat 5 set of solvers at sat competition 2016," *Proceedings of SAT Competition*, p. 28, 2016.

[36] J. Lu, Y. Liu, T. Ashur, B. Sun, and C. Li, "Improved rotational-xor cryptanalysis of simon-like block ciphers," *IET Information Security*, vol. 16, no. 4, pp. 282–300, 2022.

[37] N. Mouha, Q. Wang, D. Gu, and B. Preneel, "Differential and linear cryptanalysis using mixed-integer linear programming," in *Information Security and Cryptology: 7th International Conference, Inscrypt 2011, Beijing, China, November 30–December 3, 2011. Revised Selected Papers 7*.  Springer, 2012, pp. 57–76.

[38] N. Mouha and B. Preneel, "Towards finding optimal differential characteristics for arx: Application to salsa20," *Cryptology ePrint Archive*, 2013.

[39] L. Sun, W. Wang, and M. Wang, "More accurate differential properties of led64 and midori64," *IACR Transactions on Symmetric Cryptology*, pp. 93–123, 2018.

[40] J. Guo, G. Liu, L. Song, and Y. Tu, "Exploring sat for cryptanalysis:(quantum) collision attacks against 6-round sha-3," in *International Conference on the Theory and Application of Cryptology and Information Security*.  Springer, 2022, pp. 645–674.

[41] I. Bello, H. Pham, Q. V. Le, M. Norouzi, and S. Bengio, "Neural combinatorial optimization with reinforcement learning," 2017.

[42] M. Trimoska, S. Ionica, and G. Dequen, *Parity (XOR) Reasoning for the Index Calculus Attack*.  Springer International Publishing, 2020, p. 774–790. [Online]. Available: http://dx.doi.org/10.1007/978-3-030-58475-7_45

[43] D. Choo, M. Soos, K. M. A. Chai, and K. S. Meel, "Bosphorus: Bridging anf and cnf solvers," in *Proceedings of Design, Automation, and Test in Europe(DATE)*, 3 2019.

[44] M. Soos, K. Nohl, and C. Castelluccia, "Extending sat solvers to cryptographic problems," in *International Conference on Theory and Applications of Satisfiability Testing*.  Springer, 2009, pp. 244–257.

[45] A. Biere and M. Fleury, "Gimsatul, IsaSAT and Kissat entering the SAT Competition 2022," in *Proc. of SAT Competition 2022 – Solver and Benchmark Descriptions*, ser. Department of Computer Science Series of Publications B, T. Balyo, M. Heule, M. Iser, M. Järvisalo, and M. Suda, Eds., vol. B-2022-1.  University of Helsinki, 2022, pp. 10–11.

[46] B. Selman, H. A. Kautz, B. Cohen *et al.*, "Noise strategies for improving local search," in *AAAI*, vol. 94, 1994, pp. 337–343.

[47] J. Elffers and J. Nordström, "Divide and conquer: Towards faster pseudo-boolean solving." in *IJCAI*, vol. 18, 2018, pp. 1291–1299.

[48] A. Kyrillidis, A. Shrivastava, M. Vardi, and Z. Zhang, "Fouriersat: A fourier expansion-based algebraic framework for solving hybrid boolean constraints," in *Proceedings of the AAAI Conference on Artificial Intelligence*, vol. 34, no. 02, 2020, pp. 1552–1560.

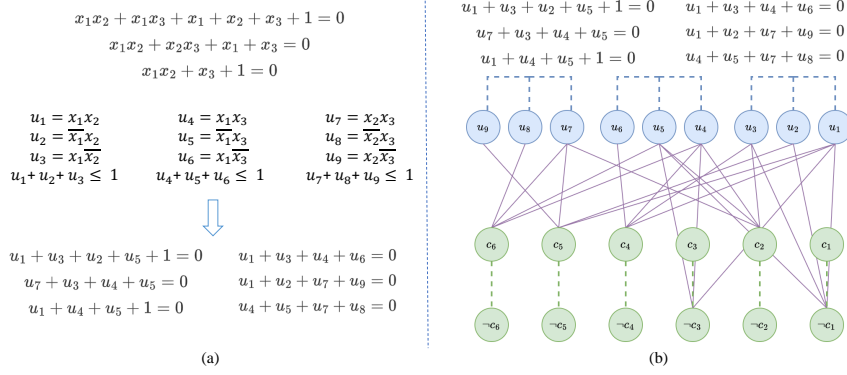

$$x_1x_2 + x_1x_3 + x_1 + x_2 + x_3 + 1 = 0$$
$$x_1x_2 + x_2x_3 + x_1 + x_3 = 0$$
$$x_1x_2 + x_3 + 1 = 0$$

$$u_1 = x_1x_2 \qquad u_4 = x_1x_3 \qquad u_7 = x_2x_3$$
$$u_2 = \overline{x_1}x_2 \qquad u_5 = \overline{x_1}x_3 \qquad u_8 = \overline{x_2}x_3$$
$$u_3 = x_1\overline{x_2} \qquad u_6 = x_1\overline{x_3} \qquad u_9 = x_2\overline{x_3}$$
$$u_1 + u_2 + u_3 \le 1 \qquad u_4 + u_5 + u_6 \le 1 \qquad u_7 + u_8 + u_9 \le 1$$

$$u_1 + u_3 + u_2 + u_5 + 1 = 0 \qquad u_1 + u_3 + u_4 + u_6 = 0$$
$$u_7 + u_3 + u_4 + u_5 = 0 \qquad u_1 + u_2 + u_7 + u_9 = 0$$
$$u_1 + u_4 + u_5 + 1 = 0 \qquad u_4 + u_5 + u_7 + u_8 = 0$$

(a)

$$u_1 + u_3 + u_2 + u_5 + 1 = 0 \qquad u_1 + u_3 + u_4 + u_6 = 0$$
$$u_7 + u_3 + u_4 + u_5 = 0 \qquad u_1 + u_2 + u_7 + u_9 = 0$$
$$u_1 + u_4 + u_5 + 1 = 0 \qquad u_4 + u_5 + u_7 + u_8 = 0$$

(b)

Figure 3: (a) A example ANF formula for changing the original ANF formula to a formula without AND operation; (b) Example ANF formula graph focusing on second-order literals

## A    Efficiency of incomplete solvers

Table 5: Comparing the efficiency of incomplete solvers for solving the MQ problem on synthetic datasets. (Average runtime: (SAT, UNSAT) ms/instance)

| Datasets | SR(5) | SR(25) | Scipher 3-8-16 | Scipher 3-16-32 | Scipher 6-8-16 | Scipher 6-16-32 | Speck 3-8-16 | Speck 6-8-16 |
|---|---|---|---|---|---|---|---|---|
| WalkSAT [46] | (3,640) | (762,744) | (4,6) | (10,12) | (289,26) | (831,899) | (39,480) | (482,538) |
| RoundingSAT [47] | (3,3) | (36758,50122) | (3,5) | (7,10) | (28,20) | (664,1801) | (23,24) | (29,35) |
| FourierSAT [48] | (1275,8670) | (9620,9687) | (983,426) | (1779,459) | (8163,416) | (8830,8862) | (8733,8689) | (8799,8912) |

## B    ANF Graph Structure focusing on second-order literals

This ANF graph structure retains only second-order literals and regards them as independent vanilla literals, representing original literals through these new literals and adding additional constraint equations. In this case, the graph has a node for every literal and every clause, an edge between every literal and each clause in which it appears, a different edge for each pair of complementary clauses, and a special edge between corresponding sets of literals.

For this type of graph, we focus on the unified representation of the vanilla literal and second-order literal. Note that we can represent all the vanilla literals with the second-order literals and their negations. For example, the vanilla literal $x_1$ can be represent as $x_1 = x_1x_2 + x_1\overline{x_2}$. Based on this idea, we introduce the negations of a second-order literal (e.g. $\overline{x_1}x_2, x_1\overline{x_2}$ of $x_1x_2$) as the new vanilla literals and we update the original ANF formula, where every three corresponding literals consist of a literal subset. Then, the ANF instance of an MQ problem with $n$ variables is represented as the ANF instance of a linear problem with $3n(n-1)$ variables.

Since the newly introduced variables are independent of each other except for the atomic clauses, to ensure that the original problem doesn't become too relaxed due to the increase in variables, we add the following two types of constraints: 1) The new vanilla literal combinations must represent the same literal in the original ANF instance; 2) At most one of the three new vanilla literals derived from the same second-order literal in the original ANF instance can be true. Notably, the second constraint is not presented in the form of an ANF formula but as an inequality, represented as a special type of edge in the graph construction. As the toy example shown in Fig. 3(a), the redefined vanilla literals $u_1 = x_1x_2$, $u_2 = \overline{x_1}x_2$, $u_3 = x_1\overline{x_2}$ and $u_4 = x_1x_3$, $u_5 = \overline{x_1}x_3$, $u_3 = x_1\overline{x_3}$ have the constraint $u_1 + u_3 + u_4 + u_6 = 0$ due to $x_1 = x_1x_2 + x_1\overline{x_2} = x_1x_3 + x_1\overline{x_3} = u_1 + u_3 = u_4 + u_6$. Besides, since $u_1, u_2, u_3$ and $u_4, u_5, u_6$ belong to the same literal subset respectively, we have $u_1 + u_2 + u_3 \le 1$ and $u_4 + u_5 + u_6 \le 1$.

Then, an undirected graph is derived from an updated ANF formula as follows and Fig. 3(b) shows the graph derived from the ANF formula in Fig. 3(a).

- Each vanilla literal $u_i$ in the ANF becomes a vertex in graph $G$.

- Each clause becomes two vertices ($\bar{c}_i$ and $c_i$) in graph $G$ according to the set of literals it contains, where $c_i$ denotes clauses with constant term 0, called the positive clause, while $\bar{c}_i$ denotes clauses with constant term 1, called the negative clause.
- An edge between literal $u_i$ and clause $c_k$ is in $g$ if and only if $x_i x_j$ appears in clause $c_k$.
- A different edge between each clause vertex $c_i$ and its complementary clause vertex $\bar{c}_i$.
- Two special edges between each literal $u_i$ and other literals in their common literal subset.

## C   Encryption Process of Scipher

Scipher has the following encryption process. Given the length $k$ of the seed key, the length $n$ of the plaintext, and the number of encryption rounds $r$, Scipher consists of multiple rounds of transformations. Each round uses a round key that is derived from the original key through specific linear transformations, with a length equal to half of the input and output lengths of that round. The input for the first round is the plaintext, and the output of the $r$-th round is the ciphertext. For the $i$-th round, we have the following round function:

$$T_1 = (L_i \lll a)\&(L_i \lll b), \ T_2 = L_i \lll c$$
$$T_3 = T_1 \oplus T_2 \oplus K_i$$
$$L_{i+1} = T_3 \oplus R_i$$
$$R_{i+1} = L_i$$

where $K_i$ represents the round key for the $i$-th round; $a$, $b$, $c$ are constants representing the number of shifted bits; $L_i$ and $R_i$ represent the high and low parts of the $i$-th round function's input, while $L_{i+1}$ and $R_{i+1}$ represent the high and low parts of the output.

## D   Limitations

The primary limitation of this paper lies in our focus on plaintext-ciphertext cryptographic Problems, while not fully exploring other cryptographic issues. On the one hand, our objective is to introduce a novel graph construction approach by leveraging the commonly encountered XOR operation in cryptographic problems, thereby proposing a graph structure based on the Arithmetic Normal Form (ANF). This allows plaintext-ciphertext cryptographic Problems to be conveniently transformed into ANF-based SAT instances. On the other hand, plaintext-ciphertext encryption algorithms are built on the hardness of the MQ problem, and thus the SAT instances derived from plaintext-ciphertext problems are also instances of the MQ problem. This provides a basis for exploring various aspects of the MQ problem. Consequently, we have chosen the widely studied plaintext-ciphertext cryptographic Problems for comprehensive decomposition and investigation. We believe that our exploration of ANF-based SAT instances will offer valuable insights and reference points for the field of SAT-based cryptanalysis. Furthermore, the proposed model can be adapted to other cryptographic problems that require the transformation of the original problem into an instance of the MQ problem.

## E   Impact Statements

The proposed CryptoANFNet, a novel graph learning framework for SAT-based cryptanalysis utilizing Arithmetic Normal Form (ANF), presents a fresh approach to tackling the challenging task of predicting plaintext-ciphertext satisfiability. Its speed improvement, surpassing heuristic solvers by a remarkable 50x and outperforming advanced learning-based SAT solvers like NeuroSAT, marks a significant advancement in cryptographic problem-solving. These strides not only enhance the efficiency and accuracy of cryptographic solutions but also open up new avenues for optimizing learning-based SAT solvers, advancing decryption algorithms, and ultimately fostering the refinement of encryption techniques.

